# A dynamical model of priming and repetition blindness

**Daphne Bavelier**
Laboratory of Neuropsychology
The Salk Institute
La Jolla, CA 92037

**Michael I. Jordan**
Department of Brain and Cognitive Sciences
Massachusetts Institute of Technology
Cambridge MA 02139

## Abstract

We describe a model of visual word recognition that accounts for several aspects of the temporal processing of sequences of briefly presented words. The model utilizes a new representation for written words, based on dynamic time warping and multidimensional scaling. The visual input passes through cascaded perceptual, comparison, and detection stages. We describe how these dynamical processes can account for several aspects of word recognition, including repetition priming and repetition blindness.

## 1   INTRODUCTION

Several psychological phenomena show that the construction of organized and meaningful representations of the visual environment requires establishing separate representations (termed episodic representations) for the different objects viewed. Three phenomena in the word recognition literature suggest that the segregation of the visual flow into separate episodic representations can be characterized in terms of specific temporal constraints. We developed a model to explore the nature of these constraints.

## 2   DESCRIPTION OF THE BEHAVIORAL DATA

In a typical priming experiment, subjects are presented with a first word, termed the "prime," and then asked to name or make a judgment to a second word, termed

the "target." The performance of subjects is compared in conditions in which the target and prime are related versus conditions in which they are unrelated.

When the prime is presented fast enough so that it cannot be identified (about 40 ms), subjects' performance on the target is facilitated when the prime and the target are identical compared to the case in which they are unrelated. This effect, known as "**masked priming**," is very short lasting, appearing only within trials, and lasting on the order of 100 ms (Humphreys, Evett, Quinlan & Besner, 1987).

If the prime, however, is presented for a period such that it is just identifiable (about 100 ms), subjects' performance on the target is hindered when prime and target are identical (Kanwisher, 1987; Humphreys et al., 1987). This effect, known as "**repetition blindness**," is conditional on the conscious identification of the prime. The size of the effect decreases as the duration between the two items increases. Repetition blindness is observed only within trials and vanishes for inter-stimulus durations on the order of 500 ms.

When the prime is presented long enough to be easily identifiable (about 250 ms or more), subjects' performance on the target is once again facilitated when prime and target are identical (Salasoo, Shiffrin & Feustel, 1985). This effect, known as "**classical repetition priming**," is long lasting, being observed not only within trials, but between trials and even between sessions. In certain experimental conditions, it has been observed to last up to a year.

These results implicate two factors influencing word recognition: the time of presentation and whether or not the prime has been identified. We have developed a model that captures the rather non-intuitive result that as the time of presentation of the prime increases, recall of the target is first facilitated, then inhibited and then facilitated again. The two main features of the model are the dynamical properties of the word representations and the dependence of the detection processes for each word on previous conscious identification of that word.

## 3   REPRESENTATION

The representation that we developed for our model is a vector space representation that allows each word to be represented by a fixed-length vector, even though the words are of different length. We developed an algorithmic method for finding the word representations that avoids some of the difficulties with earlier proposals (cf. Pinker & Prince, 1988).

The algorithm proceeds in three stages. First, *dynamic programming* (Bellman, 1957) is used to compute an inter-word similarity matrix. The transition costs in the dynamic programming procedure were based on empirically-determined values of visual similarity between individual letters (Townsend, 1971). Interestingly, we found that dynamic programming solutions naturally capture several factors that are known to be important in human sensitivity to orthographic similarity (for example, orthographic priming increases as a function of the number of letters shared between the prime and the target in a nonlinear manner, shared end-letters are more important than shared middle-letters, and relative letter position determines orthographic similarity (Humphreys et al., 1987)).

After the dynamic programming stage, *multidimensional scaling* (Torgerson, 1958) is used to convert the inter-word similarity matrix into a vector space representation in which distance correlates with similarity.

Next, word vectors are normalized by projecting them onto a semi-hypersphere. This gives the origin of the vector space a meaning, allowing us to use vector magnitude to represent signal energy.

This representation also yielded natural choices for the "blank" stimulus and the "mask" stimulus. The "blank" was taken to be the origin of the space and the "mask" was taken to be a vector on the far side of the hypersphere. In the dynamical model that we describe below, vectors that are far apart have maximally disruptive effects on each other. A distant stimulus causes the state to move rapidly away from a particular word vector, thus interfering maximally with its processing.

# 4   PROCESSING

## 4.1   FORMALIZATION OF THE PROBLEM AS A SIGNAL DETECTION PROBLEM

We formalize the problem of visual word recognition as a problem of detecting significant fluctuations of a multidimensional signal embedded in noise. This can be viewed as a maximum likelihood detection problem in which the onsets and durations of the signal are not known a priori. Our model has two main levels of processing: a perceptual stage and a detection stage.

*Perceptual Stage*

The perceptual stage is a bank of noisy linear filters. Let $W_i$ denote the $n$-dimensional word vector presented at time $t$, with components $W_{i,k}$. The word vector is corrupted with white noise $\epsilon[t]$ to form the input $u_k[t]$:

$$u_k[t] = W_{i,k} + \epsilon[t],$$

and this input is filtered:

$$r_k[t] = -a_0 r_k[t-1] - a_1 r_k[t-2] + b u_k[t] + \eta[t],$$

in the presence of additional white noise $\eta[t]$.

*Detection Stages*

The first detection stage in the model is a linear filter whose inverted impulse response is matched to the impulse response of the perceptual filter:

$$s_k[t] = -c_0 s_k[t-1] - c_1 s_k[t-2] + d r_k[t].$$

Such a filter is known as a *matched filter*, and is known to have optimality properties that make it an effective preprocessor for a system that utilizes thresholds for making decisions (van Trees, 1968). The output of the matched filter is projected onto each of the words in the lexicon to form scalar "word activation" signals $x_i[t]$ that can be compared to thresholds:

$$x_i[t] = \sum_{k=1}^{n} W_{i,k} s_k[t].$$

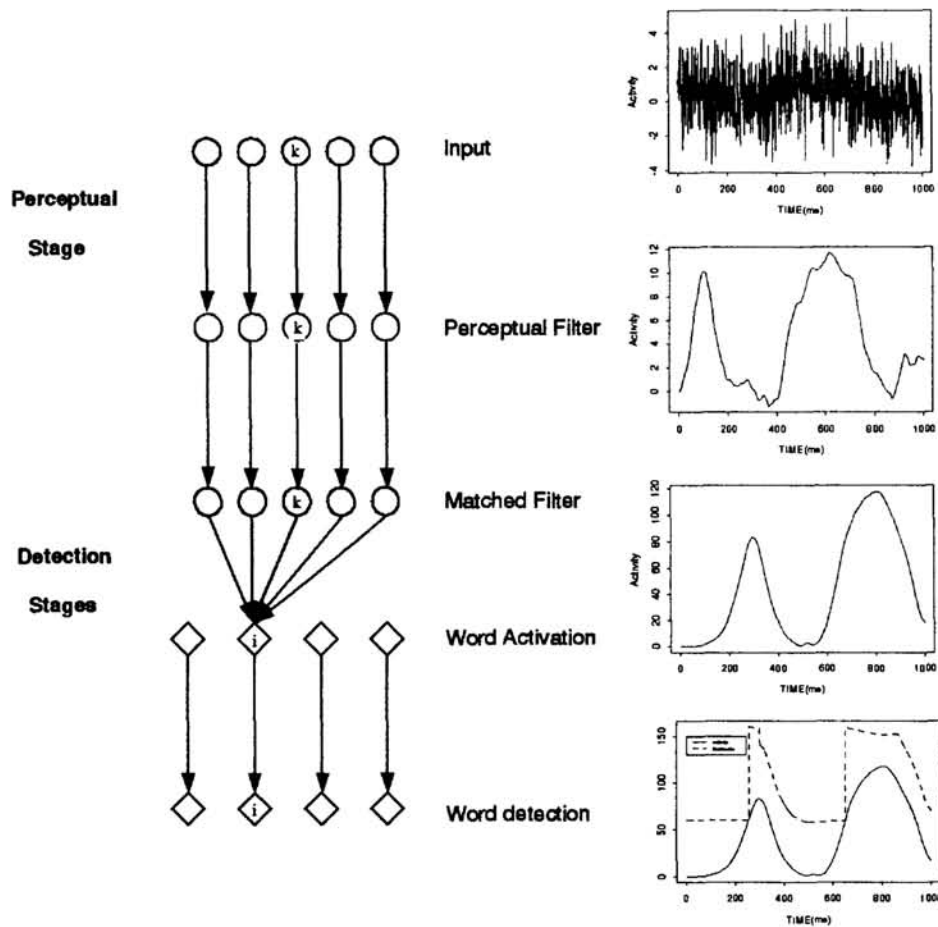

Figure 1: The processing stages of the model. The figures on the right show the signals in the model projected onto the vector for the word *bring*. *Bring* was presented for 100 ms, followed by a 300 ms blank, followed by a second presentation of *bring* for 300 ms.

The decision process is a simple binary decision based on a variable *baseline* $\mu_i[t]$ and a variable *threshold* $\theta_i[t]$:

$$y_i = \begin{cases} 1 & \text{if } x_i[t] - \mu_i[t] > \theta_i[t] \\ 0 & \text{otherwise} \end{cases}$$

## 4.2  DETECTION DYNAMICS

The problem of detecting signals that may overlap in time and have unknown onsets and unknown durations requires the system to focus on *fluctuations* rather than the absolute heights of the activation curves. Moreover, the test for significance of a fluctuation must be dependent on the state of the detection mechanism and the state of the filters. Our significance test utilizes two time-varying quantities to capture this state-dependence: the baseline $\mu$ and the threshold $\theta$.

The baseline $\mu_i[t]$ varies as follows. On time steps for which the fluctuations are subthreshold ($y_i[t] = 0$, for all $i$), each baseline simply tracks the most recent

minimum value of the corresponding word activation signal:

$$\mu_i[t] = \begin{cases} \mu_i[t-1] & \text{if } x_i[t] > \mu_i[t] \\ x_i[t] & \text{otherwise} \end{cases}$$

When a fluctuation passes threshold ($y_i[t] = 1$, for some $i$), the word $i$ is "detected," and the baselines of all words are increased:

$$\mu_k[t] = \mu_k[t-1] + \xi\phi(i,k),$$

where $\phi(i,k)$ is the angle between $W_i$ and $W_k$ and $\xi$ is a positive scaling parameter. This rule prevents multiple detections during a single presentation and it prevents the neighbors of a detected word from being detected due to their overlap with the detected word.

The threshold $\theta_i$ is subject to first-order dynamics that serve to increase or decrease the threshold as a function of the recent activation history of the word (a rudimentary form of adaptation):

$$\theta_i[t] = \alpha\theta_i[t-1] + (1-\alpha)\theta_i^0 - \beta(x_i[t] - \mu_i[t])_+,$$

where $\alpha$ and $\beta$ are positive numbers. This rule has the effect of decreasing the threshold if the activation of the word is currently above its baseline, and increasing the threshold toward its nominal value $\theta_i^0$ otherwise.

## 4.3  PARAMETERS

The parameters in the model were determined from the behavioral data and from the structural assumptions of the model in the following manner. The dynamics of the perceptual filter were determined by the time constants of masked priming, as given by the behavioral data. This choice also fixed the dynamics of the matched filter, since the matched filter was tied to the dynamics of the perceptual filter. The dynamics of the baseline $\mu$ (i.e., the value $\xi$) were determined by the constraint that a long presentation of a word not lead to multiple detections of the word. Finally, the dynamics of the threshold $\theta$ were determined by the dynamics of classical repetition priming as given by the behavioral data. Note that the behavioral data on repetition blindness were not used in adjusting the parameters of the model.

# 5   ACCOUNTS OF THE THREE BASIC PHENOMENA

## 5.1   MASKED PRIMING

The facilitation observed in masked priming is due to temporal superposition in the perceptual filter and the matched filter. At the time scale at which masked priming is observed, the activation due to the first critical word (C1) overlaps with the activation due to the second critical word (C2) (see Figure 2A), leading to a larger word activation value when C1 and C2 are identical than when they are different.

## 5.2   REPETITION BLINDNESS

The temporal superposition that leads to masked priming is also responsible for repetition blindness (see Figure 3). The temporal overlap from the filtering dynamics

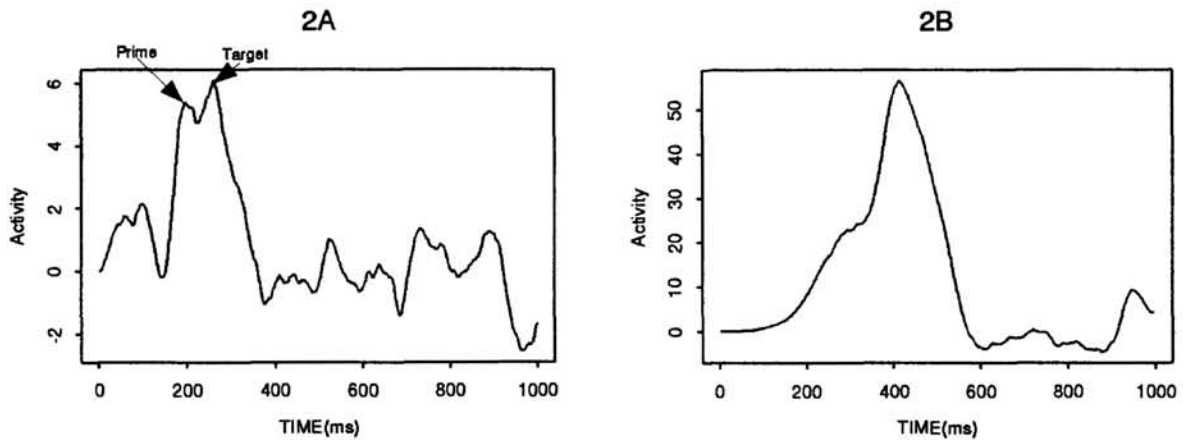

Figure 2: Activation curves at the perceptual level (A) and the matched filter level (B) for the word *bring* during the presentation of the sequence *bring, character, bring*. Each word was presented for 40 ms.

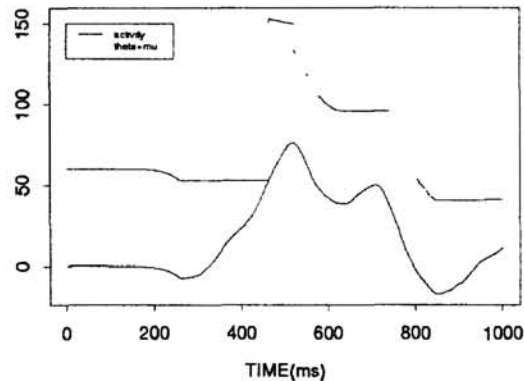

Figure 3: Activation curves for the word *bring* during the presentation of the sequence *bring, character, bring*. Each word was presented for 100 ms.

will prevent the baseline $\mu$ from getting reset to a sufficiently small value to allow a second detection. That is, repetition blindness arises because the fluctuation due to the brief presentation of C2 is not judged significant against the background of the recent detection of the word. Note that such a failure to detect the second occurrence will happen only when C1 has been correctly detected, because only then will the baseline be increased. This dependence of repetition blindness on explicit detection of the first occurrence also characterizes the behavioral data (Kanwisher, 1987).

## 5.3   CLASSICAL REPETITION PRIMING

The facilitation observed in classical repetition priming is due to the dynamics of the threshold $\theta$. The value of $\theta$ decreases during significant increases in the activation of a word; hence a smaller fluctuation in activation is needed for the next occurrence

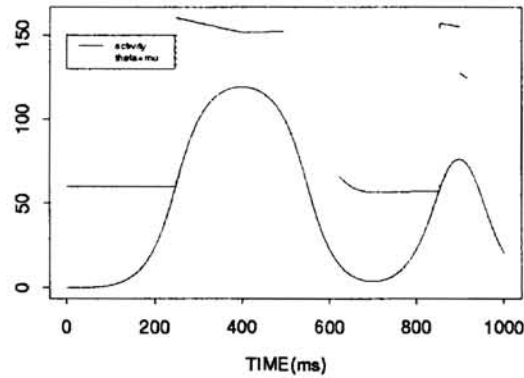

Figure 4: Activation curves for the word *bring* during the presentation of the word *bring* for 300ms, followed by a 300ms blank, followed by *bring* again for 100ms.

to be detected (see Figure 4).

# 6   OTHER DATA ACCOUNTED FOR BY THE MODEL

The model captures most of the specific characteristics of the three basic phenomena that we have reviewed. For example, it accounts for the finding of masked priming between orthographic neighbors (Humphreys, et al., 1987). This effect arises in the model because a distributed representation is used for the words. The model also captures the finding that the size of repetition blindness decreases as the interval between the critical stimuli increases (this is due to the fact that the baseline is reset to increasingly lower values as the inter-stimulus interval increases), as well as the fact that the size of repetition blindness decreases as the duration of presentation of C2 increases (because the activation for C2 continues to increase while the baseline remains fixed). Similarly, the model accounts for the finding that the manifestation of repetition blindness is dependent on the conscious identification of the first occurrence, as well the finding of repetition blindness between orthographic neighbors (Kanwisher, 1987). Specifics of classical repetition priming, such as the finding that priming is restricted to a word identity, and the fact that its size increases with the number of repetitions and diminishes as the lag between repetitions increases (Salasoo, Shiffrin & Feustel, 1985), are also captured by the model.

The model also accounts for other behavioral phenomena described in the literature on word recognition. Our vector space representation allows us to account naturally for the fact that the final words in a list are recalled better than the middle words in the list (the "recency" effect). This occurs because dissimilar words tend to have large angle between them (and therefore "inhibit" each other dynamically), whereas the "blank" is at the origin of the space and is relatively "close" to all of the words. The residual activation for a presented word therefore tends to be stronger if followed by a blank than by a dissimilar word. The model also captures certain of the effects of pattern masks on word recognition. For example, "forward" masking, a condition in which the mask precedes the word to be detected, is known to be less disruptive than "backward" masking, a condition in which the mask follows the word to be detected. This occurs in the model because of the dynamics of the

baselines: preceding a word with a mask tends to reset its baseline to lower values and therefore renders the test for significance relatively more sensitive.

# 7    CONCLUSIONS

From the point of view of the current model, the fact that the detection of repeated items is enhanced, then suppressed, then once again enhanced as the duration of the items is increased finds a natural explanation in the nature of the signal processing task that the word recognition system must solve. The signals that arrive in temporal sequence for perceptual processing have unknown onset times, unknown durations, and are corrupted by noise. The fact that signals have unknown onset times and can superimpose implies that the system must detect *fluctuations* in signal strength rather than absolute values of signal strength. The presence of noise, inevitable given the neural hardware and the complex multidimensional nature of the signal, implies that the system must detect *significant fluctuations* and must incorporate information about recent events into its significance tests. The real-time constraints of this detection task and the need to guard against errors imply that certain of the fluctuations will be missed, a fact that will result in "blindness" to repeated items at certain time scales.

### Acknowledgments

This research was funded by the McDonnell-Pew Centers for Cognitive Neuroscience at UCSD and MIT, by a grant from the McDonnell-Pew Foundation to Michael I. Jordan, and by NIDCD Grant 5RO1-DC-00128 to Helen Neville.

### References

Humphreys, G. W., Evett, L. J., Quinlan, P. T., & Besner, D. (1987). Orthographic priming. In M. Coltheart (Ed.), *Attention and Performance XII* (pp. 105-125). Hillsdale, NJ: Erlbaum.

Kanwisher, N. (1987). Repetition blindness: Type recognition without token individuation. *Cognition, 27*, 117-143.

Pinker, S. & Prince, A. (1988). On language and connectionism: Analysis of a parallel distributed processing model of language acquisition. Cognition, 28, 73-193.

Salasoo, A., Shiffrin, R. M., & Feustel, T. C. (1985). Building Permanent Memory Codes: Codification and Repetition Effects in Word Identification. *Journal of Experimental Psychology: General, 114*, 50-77.

Torgerson, W. S. (1958). Theory and Methods of Scaling. J. Wiley & Sons: New York.

Townsend, J. T. (1971). Theoritical analysis of an alphabetic confusion matrix. *Perception and Psychophysics, 9*, 40-50. (see also 449-454).

Van Trees, F. (1968). *Detection, Estimation and Modulation Theory*, Part I. New York: Wiley.